# Approximating Concavely Parameterized Optimization Problems

**Joachim Giesen**
Friedrich-Schiller-Universität Jena
Germany
joachim.giesen@uni-jena.de

**Sören Laue**
Friedrich-Schiller-Universität Jena
Germany
soeren.laue@uni-jena.de

**Jens K. Mueller**
Friedrich-Schiller-Universität Jena
Germany
jkm@informatik.uni-jena.de

**Sascha Swiercy**
Friedrich-Schiller-Universität Jena
Germany
sascha.swiercy@googlemail.com

## Abstract

We consider an abstract class of optimization problems that are parameterized concavely in a single parameter, and show that the solution path along the parameter can always be approximated with accuracy $\varepsilon > 0$ by a set of size $O(1/\sqrt{\varepsilon})$. A lower bound of size $\Omega(1/\sqrt{\varepsilon})$ shows that the upper bound is tight up to a constant factor. We also devise an algorithm that calls a step-size oracle and computes an approximate path of size $O(1/\sqrt{\varepsilon})$. Finally, we provide an implementation of the oracle for soft-margin support vector machines, and a parameterized semi-definite program for matrix completion.

## 1   Introduction

**Problem description.**   Let $D$ be a set, $I \subseteq \mathbb{R}$ an interval, and $f : I \times D \to \mathbb{R}$ such that

   (1)  $f(t, \cdot)$ is bounded from below for every $t \in I$, and

   (2)  $f(\cdot, x)$ is concave for every $x \in D$.

We study the parameterized optimization problem  $h(t) = \min_{x \in D} f(t, x)$.

A solution $x_t^* \in D$ is called optimal at parameter value $t$ if $f(t, x_t^*) = h(t)$, and $x \in D$ is called an $\varepsilon$-approximation at $t$ if $\varepsilon(t, x) := f(t, x) - h(t) \leq \varepsilon$. Of course it holds $\varepsilon(t, x_t^*) = 0$. A subset $P \subseteq D$ is called an $\varepsilon$-path if $P$ contains an $\varepsilon$-approximation for every $t \in I$. The size of a smallest $\varepsilon$-approximation path is called the $\varepsilon$-path complexity of the parameterized optimization problem.

The aim of this paper is to derive upper and lower bounds on the path complexity, and to provide efficient algorithms to compute $\varepsilon$-paths.

**Motivation.**   The rather abstract problem from above is motivated by regularized optimization problems that are abundant in machine learning, i.e., by problems of the form

$$\min_{x \in D} f(t, x) := r(x) + t \cdot l(x),$$

where $r(x)$ is a regularization- and $l(x)$ a loss term. The parameter $t$ controls the trade-off between regularization and loss. Note that here $f(\cdot, x)$ is always linear and hence concave in the parameter $t$.

**Previous work.** Due to the widespread use of regularized optimization methods in machine learning regularization path following algorithms have become an active area of research. Initially, exact path tracking methods have been developed for many machine learning problems [16, 18, 3, 9] starting with the algorithm for SVMs by Hastie et al. [10]. Exact tracking algorithms tend to be slow and numerically unstable as they need to invert large matrices. Also, the exact regularization path can be exponentially large in the input size [5, 14]. Approximation algorithms can overcome these problems [4]. Approximation path algorithms with approximation guarantees have been developed for SVMs with square loss [6], the LASSO [14], and matrix completion and factorization problems [8, 7].

**Contributions.** We provide a structural upper bound in $O(1/\sqrt{\varepsilon})$ for the $\varepsilon$-path complexity for the abstract problem class described above. We show that this bound is tight up to a multiplicative constant by constructing a lower bound in $\Omega(1/\sqrt{\varepsilon})$. Finally, we devise a generic algorithm to compute $\varepsilon$-paths that calls a problem specific oracle providing a step-size certificate. If such a certificate exists, then the algorithm computes a path of complexity in $O(1/\sqrt{\varepsilon})$. Finally, we demonstrate the implementation of the oracle for standard SVMs and a matrix completion problem. Resulting in the first algorithms for both problems that compute $\varepsilon$-paths of complexity in $O(1/\sqrt{\varepsilon})$. Previously, no approximation path algorithms have been known for standard SVMs but only a heuristic [12] and an approximation algorithm for square loss SVMs [6] with complexity in $O(1/\varepsilon)$. The best approximation path algorithm for matrix completion also has complexity in $O(1/\varepsilon)$. To our knowledge, the only known approximation path algorithm with complexity in $O(1/\sqrt{\varepsilon})$ is [14] for the LASSO.

## 2 Upper Bound

Here we show that any problem that fits the problem definition from the introduction for a compact interval $I = [a, b]$ has an $\varepsilon$-path with complexity in $O(1/\sqrt{\varepsilon})$.

Let $(a, b)$ be the interior of $[a, b]$ and let $g : (a, b) \to \mathbb{R}$ be concave, then $g$ is continuous and has a left- and right derivative $g'_-(t)$ and $g'_+(t)$, respectively, at every point $t \in I$ (see for example [15]). Note that $f(\cdot, x)$ is concave by assumption and $h$ is concave as the minimum over a family of concave functions.

**Lemma 1.** *For all* $t \in (a, b)$, $h'_-(t) \geq f'_-(t, x^*_t) \geq f'_+(t, x^*_t) \geq h'_+(t)$.

*Proof.* For all $t' < t$ it holds $h(t') \leq f(t', x^*_t)$ and hence $h(t) - h(t') \geq f(t, x^*_t) - f(t', x^*_t)$ which implies

$$h'_-(t) := \lim_{t' \uparrow t} \frac{h(t) - h(t')}{t - t'} \geq \lim_{t' \uparrow t} \frac{f(t, x^*_t) - f(t', x^*_t)}{t - t'} =: f'_-(t, x^*_t).$$

The inequality $f'_+(t, x^*_t) \geq h'_+(t)$ follows analogously, and $f'_-(t, x^*_t) \geq f'_+(t, x^*_t)$ follows after some algebra from the concavity of $f(\cdot, x^*_t)$ and the definition of the derivatives (see [15]). ☐

**Definition 2.** *Let* $I = [a, b]$ *be a compact interval,* $\varepsilon > 0$, *and* $t_0 = a$. *Let*

$$T_k = \big\{ t \, | \, t \in (t_{k-1}, b] \text{ such that } \varepsilon(t, x^*_{t_{k-1}}) := f(t, x^*_{t_{k-1}}) - h(t) = \varepsilon \big\},$$

*and* $t_k = \min T_k$ *for all integral* $k > 0$ *such that* $T_k \neq \emptyset$. *Finally, let*

$$P^* = \big\{ x^*_{t_k} \, | \, k \in \mathbb{N} \text{ such that } T_k \neq \emptyset \big\}.$$

**Lemma 3.** *Let* $s_1, \ldots, s_n \in \mathbb{R}_{>0}$, *then* $(s_1 + \ldots + s_n)(s_1^{-1} + \ldots + s_n^{-1}) \geq n^2$.

*Proof.* The claim holds for $n = 1$ as $s_1 s_1^{-1} = 1 = 1^2$. Assume the claim holds for $n - 1$ and let $a = s_1 + \ldots + s_{n-1}$ and $b = s_1^{-1} + \ldots + s_{n-1}^{-1}$. The rectangle with side lengths $as_n^{-1}$ and $bs_n$ has circumference $2(as_n^{-1} + bs_n)$ and area $as_n^{-1}bs_n = ab$. Since the square minimizes the circumference for a given area we have $2(as_n^{-1} + bs_n) \geq 4\sqrt{ab}$. The claim for $n$ now follows from

$$(a + s_n)(b + s_n^{-1}) = ab + as_n^{-1} + bs_n + 1 \geq ab + 2\sqrt{ab} + 1 = (\sqrt{ab} + 1)^2 \geq ((n-1) + 1)^2 = n^2.$$

☐

**Lemma 4.** *The size of $P^*$ is at most $\sqrt{\big((b-a)(h'_-(a) - h'_-(b))\big)/\varepsilon} \in O\big(1/\sqrt{\varepsilon}\big)$.*

*Proof.* Let $a = t_0 \leq t_1 \leq \ldots$ be the sequence from Definition 2. Define $\delta_k = t_{k+1} - t_k$ and $\Delta_k = h'_-(t_k) - h'_-(t_{k+1})$. We have

$$
\begin{aligned}
\Delta_k\, \delta_k &\geq (f'_-(t_k, x^*_{t_k}) - h'_-(t_{k+1}))(t_{k+1} - t_k) \\
&\geq \left( \frac{f(t_{k+1}, x^*_{t_k}) - f(t_k, x^*_{t_k})}{t_{k+1} - t_k} - \frac{h(t_{k+1}) - h(t_k)}{t_{k+1} - t_k} \right)(t_{k+1} - t_k) \\
&= f(t_{k+1}, x^*_{t_k}) - h(t_{k+1}) \; = \; \varepsilon(t_{k+1}, x^*_{t_k}),
\end{aligned}
$$

where the first inequality follows from Lemma 1 and the second inequality follows from concavity and the definition of derivatives (see [15]).

Thus, there exists $s_k > 0$ such that $\delta_k \geq \varepsilon s_k$ and $\Delta_k \geq s_k^{-1}$. It follows from Lemma 3 that

$$
\begin{aligned}
\varepsilon n^2 &\leq \quad \varepsilon(s_1 + \ldots + s_n)(s_1^{-1} + \ldots + s_n^{-1}) \quad \leq \quad (\delta_1 + \ldots + \delta_n)(\Delta_1 + \ldots + \Delta_n) \\
&\leq \quad (b-a)(\Delta_1 + \ldots + \Delta_n) \quad \leq \quad (b-a)(h'_-(a) - h'_-(b)),
\end{aligned}
$$

where the last inequality follows from $h'_-(b) \leq h'_-(t)$ for $t \leq b$ (which can be proved from concavity, see again [15]). Hence, the sequence $(t_k)$ and thus the size of $P^*$ must be finite, or more specifically $n$ is bounded by $\sqrt{\big((b-a)(h'_-(a) - h'_-(b))\big)/\varepsilon}$. □

**Theorem 5.** *$P^*$ is an $\varepsilon$-path for $I = [a,b]$.*

*Proof.* For any $x \in D$, $\varepsilon(\cdot, x)$ is a continuous function. Hence, $x^*_{t_k}$ is an $\varepsilon$-approximation for all $t \in [t_k, t_{k+1}]$, because if there would be $t \in (t_k, t_{k+1}]$ with $\varepsilon(t, x^*_{t_k}) > \varepsilon$, then by continuity, there would be also $t' \in (t_k, t_{k+1})$ with $\varepsilon(t, x^*_{t_k}) = \varepsilon$ which contradicts the minimality of $t_{k+1}$. The claim of the theorem follows since the proof of Lemma 4 shows that the sequence $(t_k)$ is finite and hence the intervals $[t_k, t_{k+1}]$ cover the whole $[a,b]$. □

## 3  Lower Bound

Here we show that there exists a problem that fits the problem description from the introduction whose $\varepsilon$-path complexity is in $\Omega(1/\sqrt{\varepsilon})$. This shows that the upper bound from the previous section is tight up to a constant.

Let $I = [a,b]$, $D = \mathbb{R}$, $f(t,x) = \frac{1}{2}x^2 - tx$ and thus

$$
h(t) \;=\; \min_{x \in \mathbb{R}} \left( \frac{1}{2}x^2 - tx \right) = \frac{1}{2}\big(x^*_t\big)^2 - tx^*_t = -\frac{1}{2}t^2,
$$

where the last equality follows from the convexity and differentiability of $f(t,x)$ in $x$ which together imply $\frac{\partial f}{\partial x}(t, x^*_t) = x^*_t - t = 0$.

For $\varepsilon > 0$ and $x \in \mathbb{R}$ let $I_x = \big\{ t \in [a,b] \,\big|\, \varepsilon(t,x) := \frac{1}{2}x^2 - tx + \frac{1}{2}t^2 \leq \varepsilon \big\}$, which is an interval since $\frac{1}{2}x^2 - tx + \frac{1}{2}t^2$ is a quadratic function in $t$. The length of this interval is $2\sqrt{2\varepsilon}$ independent of $x$. Hence, the $\varepsilon$-path complexity for the problem is at least $(b-a)/2\sqrt{2\varepsilon}$.

Let us compare this lower bound with the upper from the previous section which gives for the specific problem at hand, $\sqrt{\big((b-a)(h'_-(a) - h'_-(b))\big)/\varepsilon} = \sqrt{\frac{(b-a)^2}{\varepsilon}} = \frac{b-a}{\sqrt{\varepsilon}}$. Hence the upper bound is tight up to constant of at most $2\sqrt{2}$.

## 4  Generic Algorithm

So far we have only discussed structural complexity bounds for $\varepsilon$-paths. Now we give a generic algorithm to compute an $\varepsilon$-path of complexity in $O(1/\sqrt{\varepsilon})$. When applying the generic algorithm to

a specific problem a plugin-subroutine PATHPOLYNOMIAL needs to be implemented for the specific problem. The generic algorithm builds on the simple idea that has been introduced in [6] to compute an $(\varepsilon/\gamma)$-approximation (for $\gamma > 1$) and only update this approximation along the parameter interval $I = [a, b]$ when it fails to be an $\varepsilon$-approximation. The plugin-subroutine PATHPOLYNOMIAL provides a bound on the step-size for the algorithm, i.e., a certificate for how long the approximation is valid along the interval $I$. Hence we describe the idea behind the construction of this certificate first.

## 4.1 Step-size certificate and algorithm

We always consider a problem that fits the problem description from the introduction.

**Definition 6.** *Let $\mathcal{P}$ be the set of all concave polynomials $p : I \to \mathbb{R}$ of degree at most $2$. For $t \in I, x \in D$ and $\varepsilon > 0$ let*

$$\mathcal{P}_t(x, \varepsilon) := \{p \in \mathcal{P} \mid p \le h, \ f(t, x) - p(t) \le \varepsilon\},$$

*where $p \le h$ means $p(t') \le h(t')$ for all $t' \in I$.*

Note that $\mathcal{P}$ contains constant and linear polynomials with second derivative $p'' = 0$ and quadratic polynomials with constant second derivative $p'' < 0$. If $\mathcal{P}_t(x, \varepsilon) \ne \emptyset$, then $x$ is an $\varepsilon$-approximation at parameter value $t$, because there exists $p \in \mathcal{P}$ such that $\varepsilon(t, x) \le f(t, x) - p(t) \le \varepsilon$.

**Definition 7.** *[Step-size] For $t \in I = [a, b], p \in \mathcal{P}, \varepsilon > 0$, and $\gamma > 1$, let $\delta_t := t - a$ and*

$$\rho_t(p, \varepsilon) = \frac{\varepsilon}{\gamma \, \delta_t^2 \, |p''|}, \ \text{if } p'' < 0 \text{ and } \delta_t > 0.$$

*The step-size is given as*

$$\Delta_t(p, \varepsilon) \ = \ \begin{cases} \Delta_t^{(1)}(p) & : \quad p'' = 0 \\ \Delta_t^{(2)}(p, \varepsilon) & : \quad p'' < 0, \ \rho_t(p, \varepsilon) \ge \frac{1}{2} \\ \Delta_t^{(3)}(p, \varepsilon) & : \quad p'' < 0, \ \rho_t(p, \varepsilon) \le \frac{1}{2} \end{cases}$$

*where*
$$\Delta_t^{(1)}(p) \ = \ \delta_t(\gamma - 1)$$

$$\Delta_t^{(2)}(p, \varepsilon) \ = \ \sqrt{\frac{2\varepsilon}{|p''|} + \delta_t^2 \left(\rho_t(p, \varepsilon) - \frac{1}{2}\right)^2} - \delta_t \left(\rho_t(p, \varepsilon) + \frac{1}{2}\right)$$

$$\Delta_t^{(3)}(p, \varepsilon) \ = \ \sqrt{\frac{2\varepsilon}{|p''|}} \left(1 - \frac{1}{\sqrt{\gamma}}\right)$$

To simplify the notation we will skip the argument $\varepsilon$ of the step-size $\Delta_t$ whenever the value of $\varepsilon$ is obvious from the context.

**Observation 8.** *If $\rho_t(p, \varepsilon) = 1/2$, then $\Delta_t^{(2)}(p) = \Delta_t^{(3)}(p)$, because $\rho_t(p, \varepsilon) = 1/2$ implies $\delta_t = \sqrt{\frac{2\varepsilon}{\gamma \, |p''|}}$.*

**Lemma 9.** *For $t \in (a, b), x \in D, \varepsilon > 0$ and $\gamma > 1$. If there exists $p \in \mathcal{P}_t(x, \varepsilon/\gamma)$, then $x$ is an $\varepsilon$-approximation for all $t' \in [t, b]$ with $t' \le t + \Delta_t(p)$.*

*Proof.* Let $g : [a, b] \to \mathbb{R}$ be the following linear function,

$$g(t') \ = \ (t' - t)\frac{p(t) + \varepsilon/\gamma - p(a)}{t - a} + p(t) + \frac{\varepsilon}{\gamma}.$$

Then, for all $t' \in [t, b]$,

$$f(t', x) \le (t' - t)\frac{f(t, x) - f(a, x)}{t - a} + f(t, x) \le (t' - t)\frac{p(t) + \varepsilon/\gamma - p(a)}{t - a} + p(t) + \frac{\varepsilon}{\gamma} = g(t')$$

where the first inequality follows from the concavity of $f(\cdot, x)$, and the second inequality follows from $f(t, x) - p(t) \leq \varepsilon/\gamma$ and from $p(a) \leq h(a) \leq f(a, x)$. Thus, $x$ is an $\varepsilon$-approximation for all $t' \in [t, b]$ that satisfy $g(t') - p(t') \leq \varepsilon$ because

$$\varepsilon(t', x) = f(t', x) - h(t') \leq f(t', x) - p(t') \leq g(t') - p(t') \leq \varepsilon.$$

We finish the proof by considering three cases.

(i) If $p'' = 0$, then $g(t') - p(t')$ is a linear function in $t'$, and $g(t') - p(t') \leq \varepsilon$ solves to $t' - t \leq \delta_t(\gamma - 1) = \Delta_t^{(1)}(p) = \Delta_t(p)$.

(ii) If $p'' < 0$, then $g(t') - p(t')$ is a quadratic polynomial in $t'$ with second derivative $-p'' > 0$, and the equation $g(t') - p(t') \leq \varepsilon$ solves to $t' - t \leq \Delta_t^{(2)}(p)$. Note that we do not need the condition $\rho_t(p) \geq 1/2$ here.

(iii) The case $p'' < 0$ and $\rho_t(p) \leq 1/2$ can be reduced to Case (ii). From $\rho_t(p) \leq 1/2$ we obtain $t - a = \delta_t \geq \sqrt{\frac{2\varepsilon}{|p''|\gamma}}$ and thus $a \leq t - \sqrt{\frac{2\varepsilon}{|p''|\gamma}} =: \hat{a}$. Let $\hat{p}$ the restriction of $p$ onto the interval $[\hat{a}, b]$ and $\hat{\delta}_t = t - \hat{a}$, then $\hat{p}'' = p''$, and thus $\rho_t(\hat{p}) = \varepsilon/\left(\gamma \hat{\delta}_t^2 |\hat{p}''|\right) = \frac{1}{2}$. Hence by Observation 8, $\Delta_t^{(3)}(p) = \Delta_t^{(3)}(\hat{p}) = \Delta_t^{(2)}(\hat{p})$. The claim follows from Case (ii). $\qquad\square$

Assume now that we have an oracle PATHPOLYNOMIAL available that on input $t \in (a, b)$ and $\varepsilon/\gamma > 0$ returns $x \in D$ and $p \in \mathcal{P}_t(x, \varepsilon/\gamma)$, then the following algorithm GENERICPATH returns an $\varepsilon$-path if it terminates.

---

**Algorithm 1** GENERICPATH

---

**Input:** $f : [a, b] \times D \to \mathbb{R}$ that fits the problem description, and $\varepsilon > 0$
**Output:** $\varepsilon$-path for the interval $[a, b]$

choose $\hat{t} \in (a, b)$
$P := \text{COMPUTEPATH}\,(f, \hat{t}, \varepsilon)$
define $\hat{f} : [a, b] \times D \to \mathbb{R}, (t, x) \mapsto f(a + b - t, x)$ [then $\hat{f}$ also fits the problem description]
$P := P \cup \text{COMPUTEPATH}\,(\hat{f}, a + b - \hat{t}, \varepsilon)$
**return** $P$

---

**Algorithm 2** COMPUTEPATH

---

**Input:** $f : [a, b] \times D \to \mathbb{R}$ that fits the problem description, $\hat{t} \in (a, b)$ and $\varepsilon > 0$
**Output:** $\varepsilon$-path for the interval $[\hat{t}, b]$

$t := \hat{t}$ and $P := \emptyset$
**while** $t \leq b$ **do**
$\quad (x, p) := \text{PATHPOLYNOMIAL}\,\left(t, \varepsilon/\gamma\right)$
$\quad P := P \cup \{x\}$
$\quad t := \min\left\{b, t + \Delta_t(p)\right\}$
**end while**
**return** $P$

---

### 4.2 Analysis of the generic algorithm

The running time of the algorithm GENERICPATH is essentially determined by the complexity of the computed path times the cost of the oracle PATHPOLYNOMIAL. In the following we show that the complexity of the computed path is at most $O(1/\sqrt{\varepsilon})$.

**Observation 10.** *For $c \in \mathbb{R}$ let $\phi_c : \mathbb{R}_{\left[>\sqrt{|c|}\right]} \to \mathbb{R}, x \mapsto \sqrt{x^2 + c} - x$. Then we have*

1. *$\lim_{x \to \infty} \phi_c(x) = 0$*

2. *$\phi_c'(x) = \frac{x}{\sqrt{x^2 + c}} - 1$ for the derivative of $\phi_c$. Thus, $\phi_c'(x) > 0$ for $c < 0$ and $\phi_c$ is monotonously increasing.*

*Furthermore,* $\Delta_t^{(2)}(p) = \sqrt{\frac{2\varepsilon}{|p''|} + \delta_t^2 \left(\rho_t(p) - \frac{1}{2}\right)^2} - \delta_t \left(\rho_t(p) + \frac{1}{2}\right)$

$$= \sqrt{\delta_t^2 \left(\rho_t(p) + \gamma - \frac{1}{2}\right)^2 + \delta_t^2 \gamma(1-\gamma)} - \delta_t \left(\rho_t(p) + \frac{1}{2}\right)$$

$$= \sqrt{\delta_t^2 \left(\rho_t(p) + \gamma - \frac{1}{2}\right)^2 + \delta_t^2 \gamma(1-\gamma)} - \delta_t \left(\rho_t(p) + \gamma - \frac{1}{2}\right) + \delta_t(\gamma - 1)$$

$$= \phi_{\delta_t^2 \gamma(1-\gamma)} \left(\delta_t \left(\rho_t(p) + \gamma - \frac{1}{2}\right)\right) + \delta_t(\gamma - 1).$$

**Lemma 11.** *Given $t \in I$ and $p \in \mathcal{P}$, then $\Delta_t(p)$ is continuous in $|p''|$.*

*Proof.* The continuity for $|p''| > 0$ follows from the definitions of $\Delta_t^{(2)}(p)$ and $\Delta_t^{(3)}(p)$, and from Observation 8. Since $\rho_t(p) > 1/2$ for small $|p''|$ the continuity at $|p''| = 0$ follows from Observation 10, because

$$\lim_{|p''| \to 0} \Delta_t^{(2)}(p) = \lim_{|p''| \to 0} \phi_{\delta_t^2 \gamma(1-\gamma)} \left(\delta_t \cdot (\rho_t(p) + \gamma - 1/2)\right) + \delta_t(\gamma - 1) = \delta_t(\gamma - 1) = \Delta_t^{(1)}(p),$$

where we have used $\rho_t(p) \to \infty$ as $|p''| \to 0$. $\square$

**Lemma 12.** *Given $t \in I$ and $p_1, p_2 \in \mathcal{P}$, then $\Delta_t(p_1) \geq \Delta_t(p_2)$ if $|p_1''| \leq |p_2''|$.*

*Proof.* The claim is that $\Delta_t(p)$ is monotonously decreasing in $|p''|$. Since $\Delta_t$ is continuous in $|p''|$ by Lemma 11 it is enough to check the monotonicity of $\Delta_t^{(1)}(p), \Delta_t^{(2)}(p)$ and $\Delta_t^{(3)}(p)$. The monotonicity of $\Delta_t^{(1)}(p)$ and $\Delta_t^{(3)}(p)$ follows directly from the definitions of the latter. The monotonicity of $\Delta_t^{(2)}(p)$ follows from Observation 10 since we have

$$\Delta_t^{(2)}(p) = \phi_{\delta_t^2 \gamma(1-\gamma)} \left(\delta_t \left(\rho_t(p) + \gamma - \frac{1}{2}\right)\right) + \delta_t(\gamma - 1),$$

and thus $\Delta_t^{(2)}(p)$ is monotonously decreasing in $|p''|$ because $\delta_t^2 \gamma(1-\gamma) < 0$ and $\rho_t(p)$ is monotonously decreasing in $|p''|$. $\square$

**Lemma 13.** *Given $t \in I$ and $p \in \mathcal{P}$, then $\Delta_t(p)$ is monotonously increasing in $\delta_t$ and hence in $t$.*

*Proof.* Since $\Delta_t(p)$ is continuous in $\delta_t$ by Observation 8 it is enough to check the monotonicity of $\Delta_t^{(1)}(p), \Delta_t^{(2)}(p)$ and $\Delta_t^{(3)}(p)$. The monotonicity of $\Delta_t^{(1)}(p)$ and $\Delta_t^{(3)}(p)$ follows directly from the definitions of the latter. It remains to show the monotonicity of $\Delta_t^{(2)}(p)$ for $\rho_t(p) \geq \frac{1}{2}$. For $c \geq 0$ let $\phi^{-1} : \mathbb{R}_{>0} \to \mathbb{R}, y \mapsto \frac{1}{2}\left(\frac{c}{y} - y\right)$. The notation is justified because for $\phi_c^{-1}(y) > 0$ we have $\phi_c(\phi_c^{-1}(y)) = y$. Apparently, $\phi_c^{-1}$ is monotonously decreasing, and we have

$$\Delta_t^{(2)}(p) = \phi_{c_1}(\phi_{c_2}^{-1}(\delta_t)) - \delta_t = \phi_{c_1}(\phi_{c_2}^{-1}(\delta_t)) - \phi_{c_2}(\phi_{c_2}^{-1}(\delta_t)),$$

with $c_1 = \frac{2\varepsilon}{|p''|}$ and $c_2 = \frac{c_1}{\gamma}$. Note that $\phi_{c_2}^{-1}(\delta_t) > 0$ since $\rho_t(p) \geq \frac{1}{2}$, and $c_2 < c_1$ since $\gamma > 1$. Because $\phi_{c_1}' - \phi_{c_2}' < 0$ for $c_1 > c_2$, both $\phi_{c_2}^{-1}$ and $\phi_{c_1} - \phi_{c_2}$ are monotonously decreasing in their respective arguments. Hence, $\Delta_t^{(2)}(p)$ is monotonously increasing in $\delta_t$. $\square$

**Theorem 14.** *If there exists $p \in \mathcal{P}$ and $\hat{\varepsilon} > 0$ such that $|q''| \leq |p''|$ for all $q$ that are returned by the oracle PATHPOLYNOMIAL on input $t \in [a, b]$ and $\varepsilon \leq \hat{\varepsilon}$. Then Algorithm 1 terminates after at most $O\left(1/\sqrt{\varepsilon}\right)$ steps, and thus returns an $\varepsilon$-path of complexity in $O(1/\sqrt{\varepsilon})$.*

*Proof.* For all $t \in [\hat{t}, b]$, where $\hat{t} \in (a, b)$ is chosen in algorithm GENERICPATH, we have $\Delta_t(q) \geq \Delta_t(p) \geq \Delta_{\hat{t}}(p)$. Here the first inequality is due to Lemma 12 and the second inequality is due to Lemma 13. Hence, the number of steps in the first call of COMPUTEPATH is upper bounded by $(b - \hat{t})/(\min\{\Delta_{\hat{t}}(p), b - \hat{t}\}) + 1$. Similarly, the number of steps in the second call of COMPUTEPATH is upper bounded by $(\hat{t} - a)/(\min\{\Delta_{a+b-\hat{t}}(p), \hat{t} - a\}) + 1$.

For the asymptotic behavior, observe that $\Delta_{\hat{t}}(p) = \Delta_{\hat{t}}^{(1)}(p)$ does not depend on $\varepsilon$ for $p'' = 0$. For $|p''| > 0$ observe that $\lim_{\varepsilon \to 0} \rho_{\hat{t}}(p, \varepsilon) = 0$. Hence, there exists $\hat{\varepsilon} > 0$ such that $\rho_{\hat{t}}(p, \varepsilon) < 1/2$ and $\Delta_{\hat{t}}^{(3)}(p, \varepsilon) \leq b - \hat{t}$ for all $\varepsilon < \hat{\varepsilon}$, and thus

$$\frac{b - \hat{t}}{\min\{\Delta_{\hat{t}}(p), b - \hat{t}\}} + 1 = \frac{b - \hat{t}}{\Delta_{\hat{t}}^{(3)}(p)} + 1 = \sqrt{\frac{|p''|}{2\varepsilon}} \frac{\sqrt{\gamma}}{\sqrt{\gamma} - 1}(b - \hat{t}) + 1 \in O\left(\frac{1}{\sqrt{\varepsilon}}\right).$$

Analogously, $(\hat{t} - a)/(\min\{\Delta_{a+b-\hat{t}}(p), \hat{t} - a\}) + 1 \in O(1/\sqrt{\varepsilon})$, which completes the proof. $\qquad\square$

## 5   Applications

Here we demonstrate on two examples that Lagrange duality can be a tool for implementing the oracle PATHPOLYNOMIAL in the generic path algorithm. This approach obtains the step-size certificate from an approximate solution that has to be computed anyway.

### 5.1   Support vector machines

Given data points $x_i \in \mathbb{R}^d$ together with labels $y_i \in \{\pm 1\}$ for $i = 1, \ldots, n$. A support vector machine (SVM) is the following parameterized optimization problem

$$\min_{w \in \mathbb{R}^d, b \in \mathbb{R}} \left( \frac{1}{2} \|w\|^2 + t \sum_{i=1}^{n} \max\{0, 1 - y_i(w^T x_i + b)\} =: f(t, w) \right)$$

parameterized in the regularization parameter $t \in [0, \infty)$. The Lagrangian dual of the SVM is given as

$$\max_{\alpha \in \mathbb{R}^n} \left( -\frac{1}{2}\alpha^T K \alpha + \mathbf{1}^T \alpha =: d(\alpha) \right) \quad \text{s.t.} \quad 0 \leq \alpha_i \leq t, \ y^T \alpha = 0,$$

where $K = A^T A$, $A = (y_1 x_1, \ldots, y_n x_n) \in \mathbb{R}^{d \times n}$ and $y = (y_1, \ldots, y_n) \in \mathbb{R}^n$.

---
**Algorithm 3** PATHPOLYNOMIALSVM

**Input:** $t \in (0, \infty)$ and $\varepsilon > 0$
**Output:** $w \in \mathbb{R}^d$ and $p \in \mathcal{P}_t(w, \varepsilon)$

compute a primal solution $w \in \mathbb{R}^d$ and a dual solution $\alpha \in \mathbb{R}^n$ such that $f(t, w) - d(\alpha) < \varepsilon$
define $p : I \to \mathbb{R}, t' \mapsto d(\alpha t'/t)$
**return** $(w, p)$

---

**Lemma 15.** *Let $(w, p)$ be the output of* PATHPOLYNOMIALSVM *on input $t > 0$ and $\varepsilon > 0$, then $p \in \mathcal{P}_t(w, \varepsilon)$ and $|p''| \leq \max_{0 \leq \hat{\alpha} \leq 1} \hat{\alpha}^T K \hat{\alpha}$. [Hence, Theorem 14 applies here.]*

*Proof.* Let $\alpha$ be the dual solution computed by PATHPOLYNOMIALSVM and $p$ be the polynomial defined in PATHPOLYNOMIALSVM. Then,

$$p(t') = -\frac{t'^2}{t^2}\frac{1}{2}\alpha^T K \alpha + \frac{t'}{t}\mathbf{1}^T \alpha \quad \text{and thus} \quad p''(t') = -\frac{1}{t^2}\alpha^T K \alpha \leq 0$$

since $K$ is positive semidefinite. Hence, $p \in \mathcal{P}$. For $p \in \mathcal{P}_t(w, \varepsilon)$, it remains to show that $p \leq h = \min_{w \in \mathbb{R}^d} f(\cdot, w)$ and $f(t, w) - p(t) \leq \varepsilon$. The latter follows immediately from $p(t) = d(\alpha)$. For $t' > 0$ let $\alpha' = \alpha t'/t$, then $\alpha'$ is feasible for the dual SVM at parameter value $t'$ since $\alpha$ is feasible for the dual SVM at $t$. It follows, $p(t') = d(\alpha') \leq h(t') = \min_{w \in \mathbb{R}^d} f(\cdot, w)$. Finally, observe that $\alpha_i \leq t$ implies $|p''| = \frac{1}{t^2}\alpha^T K \alpha \leq \max_{0 \leq \hat{\alpha} \leq 1} \hat{\alpha}^T K \hat{\alpha}$. $\qquad\square$

The same results hold when using any positive kernel $K$. In the kernel case one has the following primal SVM (see [2]),

$$\min_{\beta \in \mathbb{R}^m, b} \left( \frac{1}{2}\beta^T K \beta + t \cdot \sum_{i=1}^{n} \max\left\{ 0, 1 - y_i \left( \sum_{j=1}^{n} \beta_j y_j K_{ij} + b \right) \right\} =: f(t, \beta) \right).$$

We have implemented the algorithm GENERICPATH for SVMs in Matlab using LIBSVM [1] as the SVM solver. To assess the practicability of the proposed algorithm we ran it on several datasets taken from the LIBSVM website. For each dataset we have measured the size of the computed $\varepsilon$-path (number of nodes) for $t \in [0.1, 10]$ and $\varepsilon \in \{2^{-i} \,|\, i = 2, \ldots, 10\}$. Figure 5.1 shows the size of paths as a function of $\varepsilon$ using double logarithmic plots. A straight line plot with slope $-\frac{1}{2}$ corresponds to an empirical path complexity that follows the function $1/\sqrt{\varepsilon}$.

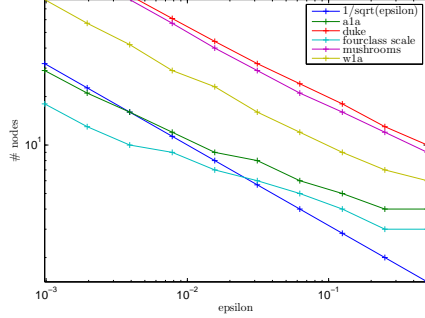

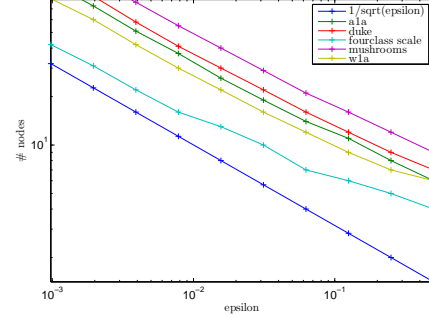

(a) Path complexity for a linear SVM

(b) Path complexity for a SVM with Gaussian kernel $\exp(-\gamma\|u - v\|_2^2)$ for $\gamma = 0.5$

## 5.2 Matrix completion

Matrix completion asks for a completion $X$ of an $(n \times m)$-matrix $Y$ that has been observed only at the indices in $\Omega \subset \{1, \ldots, m\} \times \{1, \ldots, n\}$. The problem can be solved by the following convex semidefinite optimization approach, see [17, 11, 13],

$$\min_{X \in \mathbb{R}^{n \times m}, A \in \mathbb{R}^{n \times n}, B \in \mathbb{R}^{m \times m}} \sum_{(i,j) \in \Omega} \left(X_{ij} - Y_{ij}\right)^2 + t \cdot \frac{1}{2}\big(\mathrm{tr}(A) + \mathrm{tr}(B)\big) \quad \text{s.t.} \quad \begin{pmatrix} A & X \\ X^T & B \end{pmatrix} \succeq 0.$$

The Lagrangian dual of this convex semidefinite program is given as

$$\max_{\Lambda \in \mathbb{R}^{n \times m}} - \sum_{(i,j) \in \Omega} \frac{1}{2}\Lambda_{ij}^2 + \Lambda_{ij} Y_{ij} \quad \text{s.t.} \quad \begin{pmatrix} tI & \Lambda \\ \Lambda^T & tI \end{pmatrix} \succeq 0, \text{ and } \Lambda_{ij} = 0 \text{ if } (i,j) \notin \Omega.$$

Let $f(t, \hat{X})$ for $\hat{X} = (X, A, B)$ be the primal objective function at parameter value $t$, and $d(\Lambda)$ be the dual objective function. Analogously to the SVM case we have the following:

---
**Algorithm 4** PATHPOLYNOMIALMATRIXCOMPLETION
---
**Input:** $t \in (0, \infty)$ and $\varepsilon > 0$
**Output:** $\hat{X}$ and $p \in \mathcal{P}_t(\hat{X}, \varepsilon)$

compute a primal solution $\hat{X}$ and a dual solution $\Lambda \in \mathbb{R}^{n \times m}$ such that $f(t, \hat{X}) - d(\Lambda) < \varepsilon$
define $p : I \to \mathbb{R}, t' \mapsto d\big(t'/t\,\Lambda\big)$
**return** $(\hat{X}, p)$

---

**Lemma 16.** *Let* $(\hat{X}, p)$ *be the output of* PATHPOLYNOMIALMATRIXCOMPLETION *on input* $t > 0$ *and* $\varepsilon > 0$, *then* $p \in \mathcal{P}_t(\hat{X}, \varepsilon)$ *and* $|p''| \leq \max_{\hat{\Lambda} \in \mathcal{F}_1} \|\hat{\Lambda}\|_F^2$, *where*
$$\mathcal{F}_t = \left\{ \Lambda \in \mathbb{R}^{n \times m} \,\middle|\, \begin{pmatrix} tI & \Lambda \\ \Lambda^T & tI \end{pmatrix} \succeq 0, \ \Lambda_{ij} = 0, \ \forall (i,j) \notin \Omega \right\}. \qquad \square$$

The proof for Lemma 16 is similar to the proof of Lemma 15, and Lemma 16 shows that Theorem 14 can be applied here.

**Acknowledgments** This work has been supported by a grant of the Deutsche Forschungsgemeinschaft (GI-711/3-2).

# References

[1] Chih-Chung Chang and Chih-Jen Lin. Libsvm: A library for support vector machines. *ACM Trans. Intell. Syst. Technol.*, 2(3):27:1–27:27, 2011.

[2] Olivier Chapelle. Training a Support Vector Machine in the Primal. *Neural Computation*, 19(5):1155–1178, 2007.

[3] Alexandre d'Aspremont, Francis R. Bach, and Laurent El Ghaoui. Full Regularization Path for Sparse Principal Component Analysis. In *Proceedings of the International Conference on Machine Learning (ICML)*, pages 177–184, 2007.

[4] Jerome Friedman, Trevor Hastie, Holger Höfling, and Robert Tibshirani. Pathwise Coordinate Optimization. *The Annals of Applied Statistics*, 1(2):302–332, 2007.

[5] Bernd Gärtner, Martin Jaggi, and Clement Maria. An Exponential Lower Bound on the Complexity of Regularization Paths. *arXiv.org*, arXiv:0903.4817v, 2010.

[6] Joachim Giesen, Martin Jaggi, and Sören Laue. Approximating Parameterized Convex Optimization Problems. In *Proceedings of Annual European Symposium on Algorithms (ESA)*, pages 524–535, 2010.

[7] Joachim Giesen, Martin Jaggi, and Sören Laue. Optimizing over the Growing Spectrahedron. In *Proceedings of Annual European Symposium on Algorithms (ESA)*, pages 503–514, 2012.

[8] Joachim Giesen, Martin Jaggi, and Sören Laue. Regularization Paths with Guarantees for Convex Semidefinite Optimization. In *Proceedings International Conference on Artificial Intelligence and Statistics (AISTATS)*, pages 432–439, 2012.

[9] Bin Gu, Jian-Dong Wang, Guan-Sheng Zheng, and Yue cheng Yu. Regularization Path for $\nu$-Support Vector Classification. *IEEE Transactions on Neural Networks and Learning Systems*, 23(5):800–811, 2012.

[10] Trevor Hastie, Saharon Rosset, Robert Tibshirani, and Ji Zhu. The entire regularization path for the support vector machine. *The Journal of Machine Learning Research*, 5:1391–1415, 2004.

[11] Martin Jaggi and Marek Sulovský. A Simple Algorithm for Nuclear Norm Regularized Problems. In *Proceedings of the International Conference on Machine Learning (ICML)*, pages 471–478, 2010.

[12] Masayuki Karasuyama and Ichiro Takeuchi. Suboptimal Solution Path Algorithm for Support Vector Machine. In *Proceedings of the International Conference on Machine Learning (ICML)*, pages 473–480, 2011.

[13] Sören Laue. A hybrid algorithm for convex semidefinite optimization. In *Proceedings of the International Conference on Machine Learning (ICML)*, 2012.

[14] Julien Mairal and Bin Yu. Complexity Analysis of the Lasso Regularization Path. In *Proceedings of the International Conference on Machine Learning (ICML)*, 2012.

[15] A. Wayne Roberts and Dale Varberg. *Convex functions*. Academic Press, New York, 1973.

[16] Saharon Rosset and Ji Zhu. Piecewise linear regularized solution paths. *Annals of Statistics*, 35(3):1012–1030, 2007.

[17] Nathan Srebro, Jason D. M. Rennie, and Tommi Jaakkola. Maximum-Margin Matrix Factorization. In *Proceedings of Advances in Neural Information Processing Systems 17 (NIPS)*, 2004.

[18] Zhi-li Wu, Aijun Zhang, Chun-hung Li, and Agus Sudjianto. Trace Solution Paths for SVMs via Parametric Quadratic Programming. In *KDD Worskshop: Data Mining Using Matrices and Tensors*, 2008.

